# A MODEL OF NEURAL OSCILLATOR FOR A UNIFIED SUBMODULE

A.B.Kirillov, G.N.Borisyuk, R.M.Borisyuk,
Ye.I.Kovalenko, V.I.Makarenko,V.A.Chulaevsky,
V.I.Kryukov
Research Computer Center
USSR Academy of Sciences
Pushchino, Moscow Region
142292    USSR

## ABSTRACT

A new model of a controlled neuron oscillator, proposed earlier {Kryukov et al, 1986} for the interpretation of the neural activity in various parts of the central nervous system, may have important applications in engineering and in the theory of brain functions. The oscillator has a good stability of the oscillation period, its frequency is regulated linearly in a wide range and it can exhibit arbitrarily long oscillation periods without changing the time constants of its elements. The latter is achieved by using the critical slowdown in the dynamics arising in a network of nonformal excitatory neurons {Kovalenko et al, 1984, Kryukov, 1984}. By changing the parameters of the oscillator one can obtain various functional modes which are necessary to develop a model of higher brain function.

## THE OSCILLATOR

Our oscillator comprises several hundreds of modelled excitatory neurons (located at the sites of a plane lattice) and one inhibitory neuron. The latter receives output signals from all the excitatory neurons and its own output is transmitted via feedback to every excitatory neuron (Fig. 1). Each excitatory neuron is connected bilaterally with its four nearest neighbours.

Each neuron has a threshold $r(t)$ decaying exponentially to a

value $r_\infty^e$ or $r_\infty^i$ (for an excitatory or inhibitory neuron).    A Gaussian noise with zero mean and standard deviation $\sigma$  is added to a threshold. A membrane potential of  a  neuron  is the sum of input impulses decaying exponentially when   there are  no  input.  If  the  membrane  potential  exceeds   the threshold,  the  neuron  fires  and  sends  impulses  to  the neighbouring neurons. An impulse from excitatory neuron  to excitatory one  increases  the  membrane  potential  of  the latter by $a_{ee}$, from the excitatory to the  inhibitory  -  by $a_{ei}$, and from the inhibitory to the excitatory  -  decreases the membrane potential by $a_{ie}$. We consider a  discrete  time model, the time step being equal to the absolute   refractory period.

We associate a variable $x_i(t)$ with each  excitatory   neuron. If the i-th neuron fires at step t, we take $x_i(t)=1$;  if  it does not, then $x_i(t)=0$. The mean $E(t)=1/N \; \Sigma_i x_i(t)$  will  be referred to as the network activity, where N is  the   number of excitatory neurons.

A                                    B

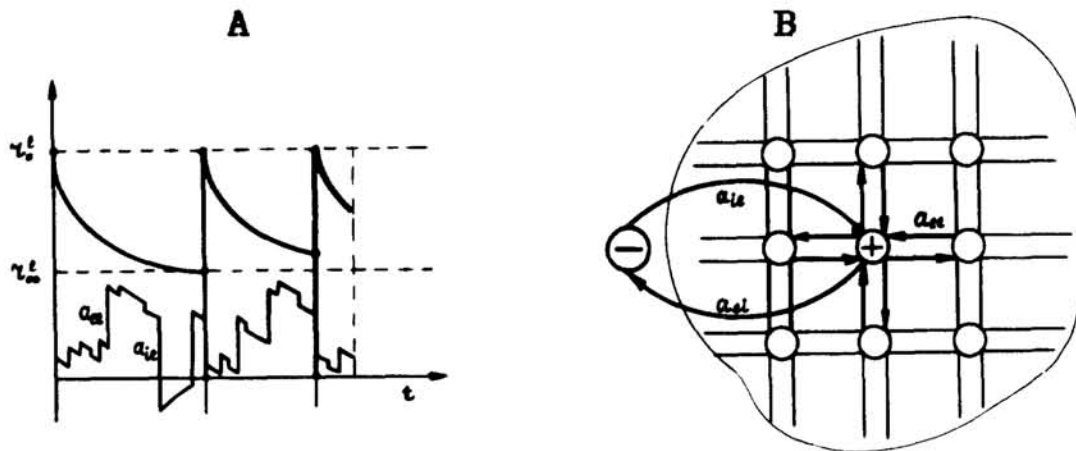

**Figure 1. A** - neuron, **B** - scheme of interconnections

Let us consider a situation when inhibitory feedback  is  cut off. Then such a model exhibits a critical  slowdown  of  the dynamics {Kovalenko et al, 1984, Kryukov, 1984}.  Namely,   if the interconnections and parameters  of  neurons  are  chosen appropriately, initial pattern of activated neurons  has  an unusually long lifetime as compared with the time of membrane potential decay. In this mode $E(t)$ is slowly   increasing  and

causes the inhibitory neuron to fire.

Now, if we turn on the negative feedback, output impulse from inhibitory neuron sharply decreases membrane potentials of excitatory neurons. As a a consequence, $E(t)$ falls down and process starts from the beginning.

We studied this oscillator by means of simulation model. There are 400 excitatory neurons (20*20 lattice) and one inhibitory neuron in our model.

## THE MAIN PROPERTIES OF THE OSCILLATOR

a. When the thresholds of excitatory neurons are high enough, the inhibitory neuron does not fire and there are no

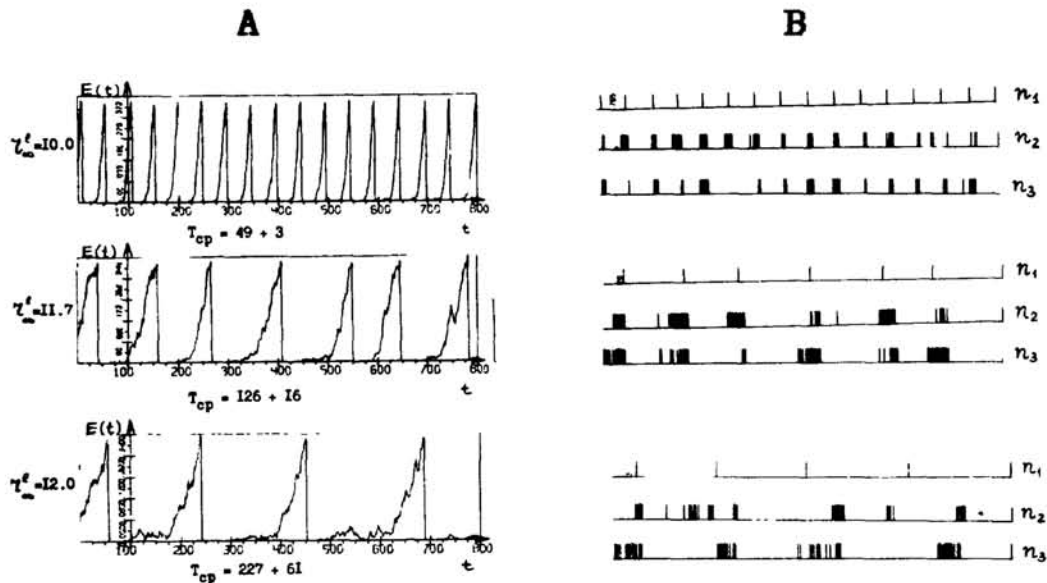

**Figure 2.** Oscillatory mode. **A** - network activity, **B** - neuron spike trains

oscillations.

b. At lower values of $r_\infty^e$ the network activity $E(t)$ changes periodically and excitatory neurons generate bursts of spikes (Fig. 2). The inhibitory neuron generates regular periodical spike trains.

c. If the parameters are chosen appropriately, the mean oscillation period is much greater than the mean interspike interval of a network neuron. The frequency of oscillations is regulated by $r_\infty^e$ (Fig. 3A) or, which is the same, by the

intensity of the input flow. The minimum period is determined by the decay rate of the inhibitory input, the maximum - by the lifetime of the metastable state.

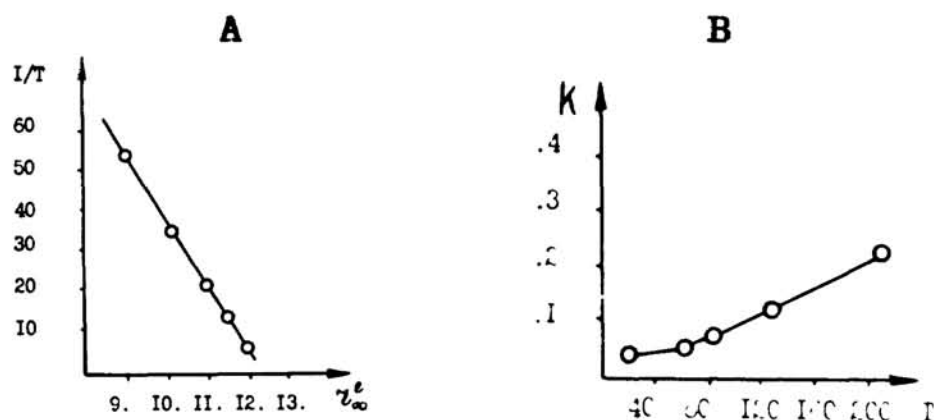

**Figure 3.** **A** - oscillation frequency **1/T** vs. threshold $r_\infty^e$,

**B** - coefficient of variation of the period **K** vs. period

d. The coefficient of variation of the period is of the order of several percent, but it increases at low frequencies (Fig. 3B). The stability of oscillations can be increased by introducing some inhomogeneity in the network, for example, when a part of excitatory neurons will receive no inhibitory signals.

## OSCILLATOR UNDER IMPULSE STIMULATION

In this section we consider first the neural network without the inhibitory neuron. But we imitate a periodic input to the network by slowly varying the thresholds **r(t)** of the excitatory neurons. Namely, we add to **r(t)** a value $\Delta r = A \cdot \sin(\omega t)$ and fire a part of the network at some phase of the sine wave. Then we look at the time needed for the network to restore its background activity. There are specific values of a phase for which this time is rather big (Fig. 4A). Now consider the full ocsillator with an oscillation period **T** (in this section T=35±2.5 time steps). We stimulate the oscillator by periodical (with the period $t_{st}$<35) sharp increase of membrane potential of each excitatory neuron by a value $a_{st}$. As the stimulation proceeds, the oscillation period gradually decreases from T=35 to some value $T_{st}$, remaining then equal to $T_{st}$. The

value of $T_{st}$ depends on the stimulation intensity $a_{st}$: as $a_{st}$ gets greater, $T_{st}$ tends to the stimulation period $t_{st}$.

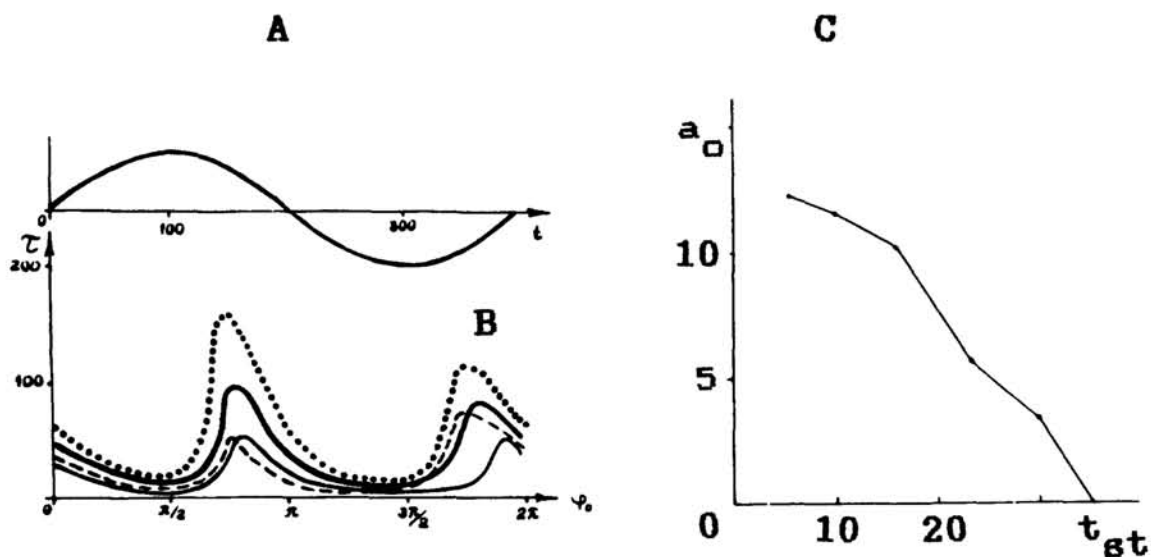

**Figure 4.** A – threshold modulation, B – duration of the network responce vs. phase of threshold modulation, C – critical stimulation intensity vs. stimulation period

For every stimulation period $t_{st}$ there is characteristic value $a_0$ of the stimulation intensity $a_{st}$, such that with $a_{st} > a_0$ the value of $T_{st}$ is equal to the stimulation period $t_{st}$. The dependence between $a_0$ and $t_{st}$ is close to a linear one (Fig. 4B). The usual relaxation oscillator also exibits a linear dependence between $a_0$ and $t_{st}$. At the same time, we did not find in our oscillator any resonance phenomena essential to a linear oscillator.

## THE NETWORK WITH INTERNAL NOISE

In a further development of the neural oscillator we tried to build a model that will be more adequate to the biological counterpart. To this end, we changed the structure of interconnections and tried to define more correctly the noise component of the input signal coming to an excitatory neuron. In the model described above we

imitated the sum of inputs from distant neurons by independent Gaussian noise. Here we used real noise produced by the network.

In order to simulate this internal noise, we randomly choose 16 distant neighbours for every exitatory neuron . Then we assume that the network elements are adjusted to work in a certain noise environment. This means that a ´mean´ internal noise would provide conditions for the neuron to be the most sensitive for the information coming from its nearest neighbors.

So, for every neuron i we calculate the sum $k_i = \sum_j x_j(t)$, where summation is over all distant neighbors of this neuron, and compare it with the mean internal noise $k = 1/N \sum_i k_i$. The internal noise for the neuron i now is $n_i = C(k_i - k)$, where $C > 0$ is a constant.

We choose model parameters in such a way that the noise component is of the order of several percent of the membrane potential. Nevertheless, the network exhibits in this case a dramatic increase of the lifetime of initial pattern of activated neurons, as compared with the network with independent Gaussian noise. A range of parameters, for which this slowdown of the dynamics is observed, is also considerably increased. Hence, longer periods and better period stability could be obtained for our generator if we use internal noise.

THE CHAIN OF THREE SUBMODULES: A MODEL OF COLUMN OSCILLATOR

Now we consider a small system constituted of three oscillator submodules, A, B and C, connected consecutively so that submodule A can transmit excitation to submodule B, B to C, and C to A. The excitation can only be transmitted when the total activity of the submodule reaches its threshold level, i.e. when the corresponding inhibitory neuron fires. After the inhibitory neuron has fired, the activity of its submodule is set to be small enough for the submodule not to be active with large probability until the excitation from another submodule comes. Therefore, we expect A, B and C to work consecutively. In fact, in our simulation experiments we observed such behavior of the

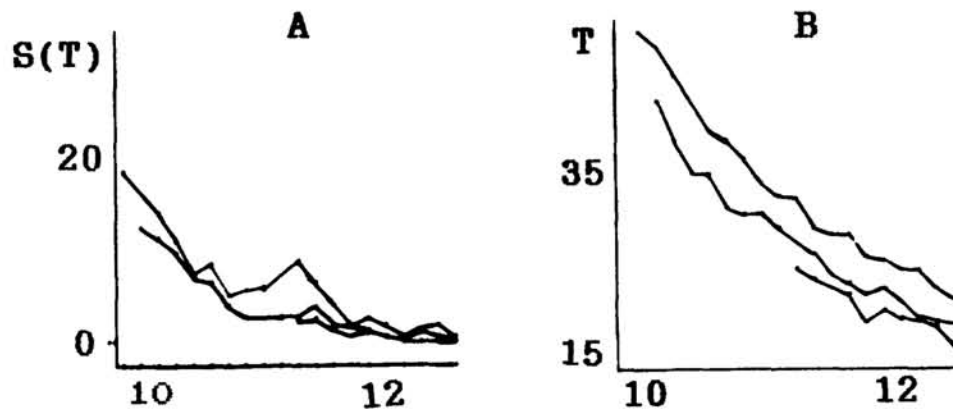

**Figure 5**. Chain of three submodules. Period of oscillations (**A**) and its standard deviation (**B**) vs. noise amplitude

closed chain of 3 basic submodules. The activity of the whole system is nearly periodic. Figure 5A displays the period $T$ vs. the noise amplitude $\sigma$. The scale of $\sigma$ is chosen so that 0.5 corresponds approximately to the resting potential. An interesting feature of the chain is that the standard deviation $S(T)$ of the period (Fig. 5B) is small enough, even for the oscillator of relatively small size. The upper lines in Fig. 5 correspond to square 10*10 network, middle – to 9*9, lower – to 8*8 one. One can see that the loss of 36 percent of elements only causes a reduction of the working range without the loss of stability.

## CONCLUSION

Though we have not considered all the interesting modes of the oscillator, we believe that, owing to the phenomenon of metastability, the same oscillator exhibits different behaviour under slightly different threshold parameters and the same and/or different inputs.

Let us enumerate the most interesting functional possibilities of the oscillator, which can be easily obtained from our results.

1. Pacemaker with the frequency regulated in a wide range and with a high period stability, as compared with the neuron (Fig. 3B).

2. Integrator (input=threshold, output=phase) with a wide

range of linear regulation (see Fig. 3A).

3.Generator of damped oscillations (for discontinuous input).

4.Delay device controlled by an external signal.

5.Phase comparator (see Fig. 4A).

We have already used these functions for the interpretation of electrical activity of several functionally different neural structures {Kryukov et al, 1986}. The other functions will be used in a system model of attention {Kryukov, 1989} presented in this volume. All these considerations justify the name of our neural oscillator – a unified submodule for a ´resonance´ neurocomputer.

## References

E. I. Kovalenko, G. N. Borisyuk, R. M. Borisyuk, A. B. Kirillov,V. I. Kryukov. Short-term memory as a metastable state. II.Simulation model, *Cybernetics and Systems Research, 2*, R. Trappl (ed.), Elsevier, pp. 266-270 (1984)

V. I. Kryukov. Short-term memory as a metastable state. I.Master equation approach, *Cybernetics and Systems Research, 2*, R. Trappl (ed.), Elsevier, pp. 261-265 (1984)

V. I. Kryukov. "Neurolocator", a model of attention (1989)(in this volume).

V. I. Kryukov, G. N. Borisyuk, R. M. Borisyuk, A. B. Kirillov, Ye. I. Kovalenko. The Metastable and Unstable States in the Brain (in Russian), Pushchino, Acad. Sci. USSR (1986) (to appear in *Stochastic Cellular Systems: Ergodicity, Memory, Morphogenesis*, Manchester University Press, 1989).
